# Cardinality Restricted Boltzmann Machines

**Kevin Swersky**      **Daniel Tarlow**      **Ilya Sutskever**
Dept. of Computer Science
University of Toronto
[kswersky,dtarlow,ilya]@cs.toronto.edu

**Ruslan Salakhutdinov**[†,‡]      **Richard S. Zemel**[†]      **Ryan P. Adams**
Dept. of Computer Science[†] and Statistics[‡]      School of Eng. and Appl. Sciences
University of Toronto      Harvard University
[rsalakhu,zemel]@cs.toronto.edu      rpa@seas.harvard.edu

## Abstract

The Restricted Boltzmann Machine (RBM) is a popular density model that is also good for extracting features. A main source of tractability in RBM models is that, given an input, the posterior distribution over hidden variables is factorizable and can be easily computed and sampled from. Sparsity and competition in the hidden representation is beneficial, and while an RBM with competition among its hidden units would acquire some of the attractive properties of sparse coding, such constraints are typically not added, as the resulting posterior over the hidden units seemingly becomes intractable. In this paper we show that a dynamic programming algorithm can be used to implement exact sparsity in the RBM's hidden units. We also show how to pass derivatives through the resulting posterior marginals, which makes it possible to fine-tune a pre-trained neural network with sparse hidden layers.

## 1   Introduction

The Restricted Boltzmann Machine (RBM) [1, 2] is an important class of probabilistic graphical models. Although it is a capable density estimator, it is most often used as a building block for deep belief networks (DBNs). The benefit of using RBMs as building blocks for a DBN is that they often provide a good initialization for feed-forward neural networks, and they can effectively utilize large amounts of unlabeled data, which has led to success in a variety of application domains [3]. Despite the benefits of this approach, there is a disconnect between the unsupervised nature of RBMs and the final discriminative task (e.g., classification) for which the learned features are used. This disconnect has motivated the search for ways to improve task-specific performance, while still retaining the unsupervised nature of the original model [4, 5]. One effective method for improving performance has been the incorporation of sparsity into the learned representation. Approaches that learn and use sparse representations have achieved good results on a number of tasks [6], and in the context of computer vision, sparsity has been linked with learning features that are invariant to local transformations [7]. Sparse features are also often more interpretable than dense representations after unsupervised learning.

For directed models, such as sparse coding [8], sparsity can be enforced using a Laplace or spike and slab prior [9]. For undirected models, introducing hard sparsity constraints directly into the energy function often results in non-trivial dependencies between hidden units that makes inference intractable. The most common way around this is to encourage sparsity during training by way of a penalty function on the expected conditional hidden unit activations given data [10]. However, this training-time procedure is a heuristic and does not guarantee sparsity at test time.

Recently, methods for efficiently dealing with highly structured global interactions within the graphical modeling framework have received considerable interest. One class of these interactions is based on assigning preferences to *counts* over subsets of binary variables [11, 12]. These are known as cardinality potentials. For example, the softmax distribution can be seen as arising from a cardinality potential that forces exactly one binary variable to be active. For general potentials over counts, it would seem that the cost of inference would grow exponentially with the number of binary variables. However, efficient algorithms have been proposed that compute exact marginals for many higher-order potentials of interest [12]. For achieving sparsity in RBMs, it turns out that a relatively simple dynamic programming algorithm by Gail et al. [13] contains the key ingredients necessary to make inference and learning efficient. The main idea behind these algorithms is the introduction of auxiliary variables that store cumulative sums in the form of a chain or a tree.

In this paper, we show how to combine these higher-order potentials with RBMs by placing a cardinality potential directly over the hidden units to form a Cardinality-RBM (CaRBM) model. This will allow us to obtain genuinely sparse representations, where only a small number of units are allowed to be active. We further show how gradients can be backpropagated through inference using a recently proposed finite-difference method [14]. On a benchmark suite of classification experiments, the CaRBM is competitive with current approaches that do not enforce sparsity at test-time.

## 2 Background

### 2.1 Restricted Boltzmann Machines

A Restricted Boltzmann Machine is a particular type of Markov random field that has a two-layer architecture, in which the visible, stochastic units $\mathbf{v} \in \{0,1\}^{N_v}$ are connected to hidden stochastic units $\mathbf{h} \in \{0,1\}^{N_h}$. The probability of the joint configuration $\{\mathbf{v}, \mathbf{h}\}$ is given by:

$$P(\mathbf{v}, \mathbf{h}) = \frac{1}{\mathcal{Z}} \exp\left(\mathbf{v}^\top W \mathbf{h} + \mathbf{v}^\top \mathbf{b}_v + \mathbf{h}^\top \mathbf{b}_h\right), \tag{1}$$

where $\mathcal{Z}$ is the normalizing constant, and $\{W \in \mathbb{R}^{N_v \times N_h}, \mathbf{b}_v \in \mathbb{R}^{N_v}, \mathbf{b}_h \in \mathbb{R}^{N_h}\}$ are the model parameters, with $W$ representing visible-to-hidden symmetric interaction terms, and $\mathbf{b}_v, \mathbf{b}_h$ representing visible and hidden biases respectively. The derivative of the log-likelihood with respect to the model parameters[1] $W$ can be obtained from Eq. 1:

$$\frac{\partial \log P(\mathbf{v}; \theta)}{\partial W} = \mathbb{E}_{P_{\text{data}}}[\mathbf{v}\mathbf{h}^\top] - \mathbb{E}_{P_{\text{model}}}[\mathbf{v}\mathbf{h}^\top], \tag{2}$$

where $\mathbb{E}_{P_{\text{data}}}[\cdot]$ denotes an expectation with respect to the data distribution

$$P_{\text{data}}(\mathbf{h}, \mathbf{v}; \theta) = P(\mathbf{h} \,|\, \mathbf{v}; \theta) \, P_{\text{data}}(\mathbf{v}), \tag{3}$$

where $P_{\text{data}}(\mathbf{v}) = \frac{1}{N} \sum_n \delta(\mathbf{v} - \mathbf{v}_n)$ represents the empirical distribution, and $\mathbb{E}_{P_{\text{model}}}[\cdot]$ is an expectation with respect to the distribution defined by the model, as in Eq. 1. Exact maximum likelihood learning in this model is intractable because exact computation of the expectation $\mathbb{E}_{P_{\text{model}}}[\cdot]$ takes time that is exponential in the number of visible or hidden units. Instead, learning can be performed by following an approximation to the gradient, the "Contrastive Divergence" (CD) objective [15].

After learning, the hidden units of the RBM can be thought of as features extracted from the input data. Quite often, they are used to initialize a deep belief network (DBN), or they can be used directly as inputs to some other learning system.

### 2.2 The Sparse RBM (SpRBM)

For many challenging tasks, such as object or speech recognition, a desirable property for the hidden variables is to encode the data using sparse representations. That is, given an input vector $\mathbf{v}$, we would like the corresponding distribution $P(\mathbf{h}|\mathbf{v})$ to favour sparse configurations. The resulting features are often more interpretable and tend to improve performance of the learning systems that use these features as inputs. On its own, it is highly unlikely that the RBM will produce sparse features. However, suppose we have some desired target expected sparsity $\rho$. If $q_j$ represents a

running average of the hidden unit marginals $q_j = 1/N \sum_n P(h_j = 1|\mathbf{v}_n)$, then we can add the following penalty term to the log-likelihood objective [16]:

$$\lambda \left( \rho \log q_j + (1 - \rho) \log(1 - q_j) \right), \tag{4}$$

where $\lambda$ represents the strength of the penalty. This penalty is proportional to the negative of the KL divergence between the hidden unit marginals and the target sparsity probability. The derivative with respect to the activity on any case n is proportional to $\lambda(\rho - q_j)$. Note that this is applied to each hidden unit independently and has the intuitive property of encouraging each hidden unit to activate with proportion $\rho$ across the dataset.

If the hidden unit activations are stored in a matrix where each row corresponds to a training example, and each column corresponds to a hidden unit, then this is enforcing sparsity in the columns of the matrix. This is also referred to as *lifetime* sparsity. When using the SpRBM model, the hope is that each individual example will be encoded by a sparse vector, corresponding to sparsity across the rows, or *population* sparsity.

## 3 The Cardinality Potential

Consider a distribution of the form

$$q(\mathbf{x}) = \frac{1}{\mathcal{Z}} \psi \left( \sum_{j=1}^{N} x_j \right) \prod_{j=1}^{N} \phi_j(x_j), \tag{5}$$

where $\mathbf{x}$ is a binary vector and $\mathcal{Z}$ is the normalizing constant. This distribution consists of non-interacting terms, with the exception of the $\psi(\cdot)$ potential, which couples all of the variables together. This is a cardinality potential (or "counts potential"), because it depends only on the number of 1's in the vector $\mathbf{x}$, but not on their identity. This distribution is useful for imposing sparsity because it allows us to represent the constraint that the vector $\mathbf{x}$ can have at most $k$ elements set to one.

There is an efficient exact inference algorithm for computing the normalizing constant and marginals of this distribution. This can be interpreted as a dynamic programming algorithm [13, 17], or as an instance of the sum-product algorithm [18]. We prefer the sum-product interpretation because it makes clear how to compute marginal distributions over binary variables, how to compute marginal distributions over total counts, and how to draw an exact joint sample from the model (pass messages forwards, then sample backwards) and also lends itself towards extensions. In this view, we create $N$ auxiliary variables $z_j \in \{1, \ldots, N\}$. The auxiliary variables are then deterministically related to the $\mathbf{x}$ variables by setting $z_j = \sum_{k=1}^{j} x_k$, where $z_j$ represents the cumulative sum of the first $j$ binary variables.

More formally, consider the following joint distribution $\hat{q}(\mathbf{x}, \mathbf{z})$:

$$\hat{q}(\mathbf{x}, \mathbf{z}) = \prod_{j=1}^{N} \phi_j(x_j) \cdot \prod_{j=2}^{N} \gamma(x_j, z_j, z_{j-1}) \cdot \psi(z_N). \tag{6}$$

We let $\gamma(x_j, z_j, z_{j-1})$ be a deterministic "addition potential", which assigns the value one to any triplet $(\mathbf{x}, \mathbf{z}, \mathbf{z}')$ satisfying $\mathbf{z} = \mathbf{x} + \mathbf{z}'$ and zero otherwise. Note that the second product ranges from $j = 2$, and that $z_1$ is replaced with $x_1$. This notation represents the observation that $z_j$ can be computed either as $z_j = \sum_{k=1}^{j} x_k$, or more simply as $z_j = z_{j-1} + x_j$. The latter is preferable, because it induces a chain-structured dependency graph amongst the $\mathbf{z}$ and $\mathbf{x}$ variables. Thus, the distribution $\hat{q}(\mathbf{x}, \mathbf{z})$ has two important properties. First, it is chain-structured, and therefore we can perform exact inference using the sum-product algorithm. By leveraging the fact that at most $k$ are allowed to be on, the runtime can be made to be $O(Nk)$ by reducing the range of each $z_i$ from $\{1, \ldots, N\}$ to $\{1, \ldots, k + 1\}$. Second, the posterior $\hat{q}(\mathbf{z}|\mathbf{x})$ assigns a probability of 1 to the configuration $\mathbf{z}^*$ that is given by $z_j^* = \sum_{k=1}^{j} x_j$ for all $j$. This is a direct consequence of the sum-potentials $\gamma(\cdot)$ enforcing the constraint $z_j^* = x_j + z_{j-1}^*$. Since $z_N^* = \sum_{j=1}^{N} x_j$, it follows that $q(\mathbf{x}) = \hat{q}(\mathbf{x}, \mathbf{z}^*)$, and since $q(\mathbf{z}|\mathbf{x})$ concentrates all of its mass on $\mathbf{z}^*$, we obtain:

$$\hat{q}(\mathbf{x}) = \sum_{\mathbf{z}} \hat{q}(\mathbf{x}, \mathbf{z}) = \sum_{\mathbf{z}} \hat{q}(\mathbf{z}|\mathbf{x})\hat{q}(\mathbf{x}) = \hat{q}(\mathbf{x}, \mathbf{z}^*) = q(\mathbf{x}). \tag{7}$$

This shows that $q(\mathbf{x})$ is the marginal distribution of the chain-structured distribution $\hat{q}(\mathbf{x}, \mathbf{z})$. By running the sum-product algorithm on $\hat{q}$ we can recover the singleton marginals $\mu_j(x_j)$, which are also the marginals of $q(\cdot)$. We can likewise sample from $q$ by computing all of the pairwise marginals $\mu_{j+1,j}(z_{j+1}, z_j)$, computing the pairwise conditionals $\mu_{j+1,j}(z_{j+1}|z_j)$, and sampling each $z_j$ sequentially, given $z_{j-1}$, to obtain a sample $\mathbf{z}$. The vector $\mathbf{x}$ can be recovered via $x_j = z_j - z_{j-1}$. The basic idea behind this algorithm is given in [13] and the sum-product interpretation is elaborated upon in [18].

There are many algorithmic extensions, such as performing summations in tree-structured distributions, which allow for more efficient inference with very large $N$ (e.g. $N > 1000$) using fast Fourier transforms [19, 18]. But in this work we only use the chain-structured distribution $\hat{q}$ described above with the restriction that there are only $k$ states.

## 4 The Cardinality RBM (CaRBM)

The Cardinality Restricted Boltzmann Machine is defined as follows:

$$P(\mathbf{v}, \mathbf{h}) = \frac{1}{\mathcal{Z}} \exp\left(\mathbf{v}^\top W \mathbf{h} + \mathbf{v}^\top \mathbf{b}_v + \mathbf{h}^\top \mathbf{b}_h\right) \cdot \psi_k\left(\sum_{j=1}^{N_h} h_j\right), \tag{8}$$

where $\psi_k$ is a potential given by $\psi_k(c) = 1$ if $c \leq k$ and 0 otherwise. Observe that the conditional distribution $P(\mathbf{h}|\mathbf{v})$ assigns a non-zero probability mass to a vector $\mathbf{h}$ only if $|\mathbf{h}| \leq k$. The cardinality potential implements competition in the hidden layer because now, a data vector $\mathbf{v}$ can be explained by at most $k$ hidden units. This form of competition is similar to sparse coding in that there may be many non-sparse configurations that assign high probability to the data, however only sparse configurations are allowed to be used. Unlike sparse coding, however, the CaRBM learning problem involves maximizing the likelihood of the training data, rather than minimizing a reconstruction cost. Using the techniques from the previous section, computing the conditional distribution $P(\mathbf{h}|\mathbf{v})$ is tractable, allowing us to use learning algorithms like CD or stochastic maximum likelihood [20]. The conditional distribution $P(\mathbf{v}|\mathbf{h})$ is still factorial and easy to sample from.

Perhaps the best way to view the effect of the cardinality potential is to consider the case of $k = 1$ with the further restriction that configurations with 0 active hidden units are disallowed. In this case, the CaRBM reduces to an ordinary RBM with a single multinomial hidden unit. A similar model to the CaRBM is the Boltzmann Perceptron [21], which also introduces a term in the energy function to promote competition between units; however, they do not provide a way to efficiently compute marginals or draw joint samples from $P(\mathbf{h}|\mathbf{v})$. Another similar line of work is the Restricted Boltzmann Forest [22], which uses $k$ groups of multinomial hidden units.

We should note that the actual marginal probabilities of the hidden units given the visible units are not guaranteed to be sparse, but rather the distribution assigns zero mass to any hidden configuration that is not sparse. In practice though, we find that after learning, the marginal probabilities do tend to have low entropy. Understanding this as a form of regularization is a topic left for future work.

### 4.1 The Cardinality Marginal Nonlinearity

One of the most common ways to use an RBM is to consider it as a pre-training method for a deep belief network [2]. After one or several RBMs are trained in a greedy layer-wise fashion, the network is converted into a deterministic feed-forward neural network that is fine-tuned with the backpropagation algorithm. The fine-tuning step is important for getting the best results with a DBN model [23]. While it is easy to convert a stack of standard RBMs into a feed-forward neural network, turning a stack of CaRBMs into a feed-forward neural network is less obvious, because it is not clear what nonlinearity should be used.

Observe that in the case of a standard, binary-binary RBM, the selected nonlinearity is the sigmoid $\sigma(\mathbf{x}) \equiv 1/(1+\exp(-\mathbf{x}))$. We can justify this choice by noticing that it is the expectation of the conditional distribution $P(\mathbf{h}|\mathbf{v})$, namely

$$\sigma(W^\top \mathbf{v} + \mathbf{b}_h) = \mathbb{E}_{P(\mathbf{h}|\mathbf{v})}[\mathbf{h}], \tag{9}$$

where the sigmoid is applied to the vector in an element-wise fashion. In particular, using the conditional expectation as the nonlinearity is a fundamental ingredient in the variational lower bound that justifies the greedy layer-wise procedure [2]. It also appears naturally when the score matching estimator is applied to RBMs over Gaussian-distributed visible units [24, 25]. This justification suggests that for the CaRBM, we should choose a nonlinearity $\mu(\cdot)$ which will satisfy the following equality:

$$\mu(W^\top \mathbf{v} + \mathbf{b}_h) = \mathbb{E}_{P(\mathbf{h}|\mathbf{v})}[\mathbf{h}], \tag{10}$$

where the conditional $P(\mathbf{h}|\mathbf{v})$ can be derived from Eq. 8. First note that such a nonlinear function exists, because the distribution $P(\mathbf{h}|\mathbf{v})$ is completely determined by the total input $W^\top \mathbf{v} + \mathbf{b}_h$. Therefore, the feed-forward neural network that is obtained from a stack of CaRBMs uses a message-passing algorithm to compute the nonlinearity $\mu(\cdot)$. We should note that $\mu$ depends on $k$, the number of units that can take on the value 1, but this is a constant that is independent of the input. In practice, we keep $k$ fixed to the $k$ that was used in unsupervised training.

To compute gradients for learning the network, it is necessary to "backpropagate" through $\mu$, which is equivalent to multiplying by the Jacobian of $\mu$. Analytic computation of the Jacobian, however, results in an overly expensive $O(N^2)$ algorithm. We also note that it is possible to manually differentiate the computational graph of $\mu$ by passing the derivatives back through the sum-product algorithm. While this approach is correct, it is difficult to implement and can be numerically unstable.

We propose an alternative approach to multiplying by the Jacobian of $\mu$. Let $\mathbf{x} = W^\top \mathbf{v} + \mathbf{b}_h$ be the total input to the RBM's hidden units, then the Jacobian $J(\mathbf{x})$ is given by:

$$\begin{aligned} J(\mathbf{x}) &= \mathbb{E}_{P(\mathbf{h}|\mathbf{v})}[\mathbf{h}\mathbf{h}^\top] - \mathbb{E}_{P(\mathbf{h}|\mathbf{v})}[\mathbf{h}]\, \mathbb{E}_{P(\mathbf{h}|\mathbf{v})}[\mathbf{h}^\top], \\ &= \mathbb{E}_{P(\mathbf{h}|\mathbf{v})}[\mathbf{h}\mathbf{h}^\top] - \mu(\mathbf{x})\mu(\mathbf{x})^\top. \end{aligned} \tag{11}$$

We need to multiply by the transpose of the Jacobian from the right, since by the chain rule,

$$\frac{\partial L}{\partial \mathbf{x}} = \frac{\partial \mu}{\partial \mathbf{x}}^\top \frac{\partial L}{\partial \mu} = J(\mathbf{x})^\top \frac{\partial L}{\partial \mu}, \tag{12}$$

where $L$ is the corresponding loss function. One way to do this is to reuse the sample $\mathbf{h} \sim P(\mathbf{h}|\mathbf{v})$ in order to obtain a rank-one unbiased estimate of $\mathbb{E}_{P(\mathbf{h}|\mathbf{v})}[\mathbf{h}\mathbf{h}^\top]$, but we found this to be inaccurate. Luckily, Domke [14] makes two critical observations. First, the Jacobian $J(\mathbf{x})$ is symmetric (see Eq. 11). Second, it is easy to multiply by the Jacobian of any function using numerical differentiation, because multiplication by the Jacobian (without a transpose) is precisely a directional derivative.

More formally, let $f(\mathbf{x})$ be any differentiable function and $J$ be its Jacobian. For any vector $\ell$, it can be easily verified that:

$$\lim_{\epsilon \to 0} \frac{f(\mathbf{x} + \epsilon\ell) - f(\mathbf{x})}{\epsilon} = \lim_{\epsilon \to 0} \frac{f(\mathbf{x}) + \epsilon J\ell + o(\epsilon) - f(\mathbf{x})}{\epsilon} = \lim_{\epsilon \to 0} \frac{o(\epsilon)}{\epsilon} + \frac{\epsilon J\ell}{\epsilon} = J\ell. \tag{13}$$

Since $\mu$ is a differentiable function, we can compute $J(\mathbf{x})\ell$ by a finite difference formula:

$$J(\mathbf{x})\ell \approx \frac{\mu(\mathbf{x} + \epsilon\ell) - \mu(\mathbf{x} - \epsilon\ell)}{2\epsilon}. \tag{14}$$

Using the symmetry of the Jacobian of $\mu$, we can backpropagate a vector of derivatives $\partial L/\partial \mu$ using Eq. 14. Of the approaches we tried, we found this approach to provide the best combination of speed and accuracy.

## 5   Experiments

The majority of our experiments were carried out on various binary datasets from Larochelle et al [26], hence referred to as the Montreal datasets. Each model was trained using the CD-1 algorithm with stochastic gradient descent on mini-batches. For training the SpRBM, we followed the guidelines from Hinton [27].

## 5.1 Training CaRBMs

One issue when training a model with lateral inhibition is that in the initial learning epochs, a small group of hidden units can learn global features of the data and effectively suppress the other hidden units, leading to "dead units". This effect has been noted before in energy-based models with competition [22]. One option is to augment the log-likelihood with the KL penalty given in Eq. 4. In the SpRBM, this penalty term is used to encourage each hidden unit to be active a small number of times across the training set, which indirectly provides sparsity per-example. In the CaRBM it is used to ensure that each hidden unit is used roughly equally across the training set, while the per-example sparsity is directly controlled.

We observed that dead units occurred only with a random initialization of the parameters and that this was no longer an issue once the weights had been properly initialized. In our experiments, we used the KL penalty during unsupervised learning, but not during supervised fine-tuning.

A related issue with SpRBMs is that if the KL penalty is set too high then it can create dead examples (examples that activate no hidden units). Note that the KL penalty will not penalize this case as long as the inter-example activations matches the target probability $\rho$.

## 5.2 Comparing CaRBM with SpRBM

Both the CaRBM and SpRBM models attempt to achieve the same goal of sparsity in the hidden unit activations. However, the way in which they accomplish this is fundamentally different.

For datasets such as MNIST, we found the two models to give qualitatively similar results. Indeed, this seemed to be the case for several datasets. On the convex dataset, however, we noticed that the models produced quite different results. The convex dataset consists of binary $28 \times 28$-pixel images of polygons (sometimes with multiple polygons per image). Figure 1 (a) shows several examples from this dataset. Unlike the MNIST dataset, there is a large variation in the number of active pixels in the inputs. Figure 1 (e) shows the distribution of the number of pixels taking the value 1. In some examples, barely any pixels are active, while in others virtually every pixel is on.

For both models, we set the target sparsity to $10\%$. We next performed a grid search over the strength of the KL penalty until we found a setting that achieved an average hidden unit population sparsity that matched the target without creating dead examples (in the case of the SpRBM) or dead units (in the case of the CaRBM). Figure 1 (d) and (h) show that both models achieve the desired mean population sparsity. However, the SpRBM exhibits a heavy-tailed distribution over activations, with some examples activating over half of the hidden units. By comparison, all inputs activate the maximum number of allowable hidden units in the CaRBM, generating a spike at $10\%$. Indeed, in the CaRBM, the hidden units suppress each other through competition, while in the SpRBM there is no such direct competition. Figure 1 (b) and (f) display the learned weights. Both models appear to give qualitatively similar results, although the CaRBM weights appear to model slightly more localized features at this level of sparsity.

## 5.3 Classification Performance

To evaluate the classification performance of CaRBMs, we performed a set of experiments on the Montreal datasets. We conducted a random search over hyperparameter settings as recommended by Bergstra & Bengio [28], and set the target sparsity to be between $2.5\%$ and $10\%$. Table 1 shows that the CarBM and SpRBM achieve comparable performance. On this suite we found that the validation sets were quite small and prone to overfitting. For example, both the SpRBM and CaRBM achieve $0.5\%$ validation error on the rectangles dataset. Interestingly, for the convex dataset, the SpRBM model, chosen by cross-validation, used a weak penalty strength and only achieved a population sparsity of $25\%$. As we increased the strength of the sparsity penalty, classification performance in the SpRBM degraded, but the desired sparsity level was still not achieved.

## 5.4 CIFAR-10 Patches

We extracted $16 \times 16$ whitened image patches from the CIFAR-10 dataset [29] and trained both models. Figure 2 (a) shows learned filters of the CaRBM model (both models behave similarly

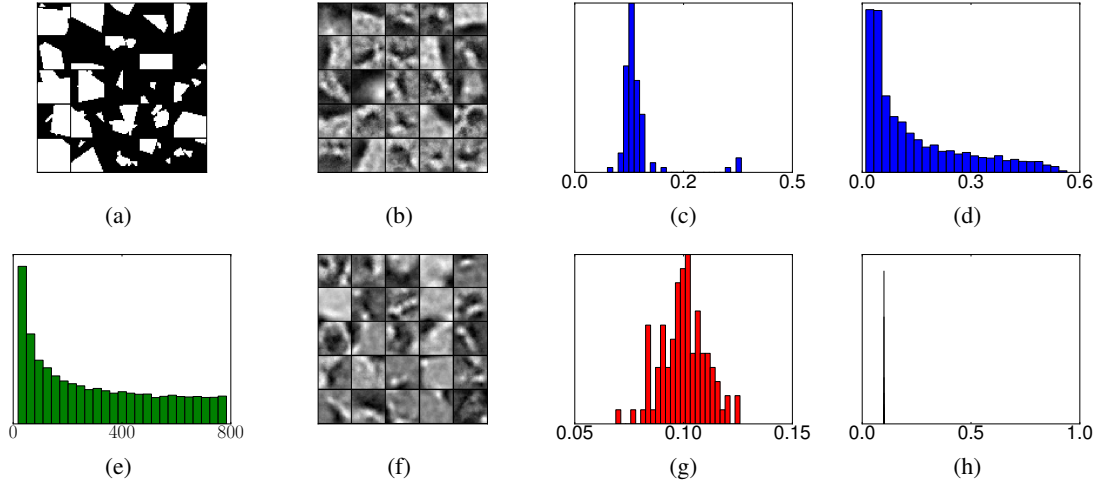

Figure 1: (a),(e) Samples from the Convex dataset and the distribution of the number of pixels in each image with the value 1. (b),(f) Visualization of the incoming weights to 25 randomly selected hidden units in the SpRBM and CaRBM models respectively. (c),(g) The distribution of the mean lifetime activations (across examples) of the hidden units in the SpRBM and CaRBM respectively. (d),(h) The distribution of the mean population activations (within examples) of the hidden units in the SpRBM and CaRBM respectively.

| Dataset | RBM | SpRBM | CaRBM | Dataset | RBM | SpRBM | CaRBM |
|---|---|---|---|---|---|---|---|
| rectangles | 4.05% | 2.66% | 5.60% | convex | 20.66% | 18.52% | 21.13% |
| background im | 23.78% | 23.49% | 22.16% | mnist basic | 4.42% | 3.84% | 3.65% |
| background im rot | 58.21% | 56.48% | 56.39% | mnist rot | 14.83% | 13.11% | 12.40% |
| recangles im | 24.24% | 22.50% | 22.56% | background rand | 12.96% | 12.97% | 12.67% |

Table 1: Test-set classification errors on the Montreal datasets.

and so we just display the CaRBM weights). Observe that the learned weights resemble Gabor-like filters. These features are often considered to be beneficial for classification when modeling images.

## 5.5   Topic Modeling with the NIPS Dataset

One form of data with highly variable inputs is text, because some words are used much more frequently than others. We applied the SpRBM and CaRBM to the NIPS dataset[2], which consists of 13649 words and 1740 papers from NIPS conferences from 1987 to 1999. Each row corresponds to a paper, each column corresponds to a word, and the entries are the number of times each word appears in each paper. We binarized the dataset by truncating the word counts and train the SpRBM and CaRBM models with 50 hidden units, searching over learning rates and KL penalty strengths until 10% sparsity is achieved without dead units or examples. Once a model is learned, we define a topic for a hidden unit by considering the 5 words with the highest connections to that unit. We conjecture that sparse RBMs should be beneficial in learning interpretable topics because there will be fewer ways for hidden units to collude in order to model a given input.

Table 2 shows the result of picking a general topic and finding the closest matching hidden unit from each model. While all models discover meaningful topics, we found that the grouping of words produced by the RBM tended to be less cohesive than those produced by the SpRBM or CaRBM. For example, many of the hidden units contain the words 'abstract' and 'reference', both of which appear in nearly every paper.

Figure 2 (b)-(d) displays the effect that the KL penalty $\lambda$ has on the population sparsity of the SpRBM. For a fairly narrow range, if $\lambda$ is too small then the desired sparsity level will not be met.

| Model | Computer Vision | Neuroscience | Bayesian Inference |
|---|---|---|---|
| RBM | images, pixel, computer, quickly, stanford | inhibitory, organization, neurons, synaptic, explain | probability, bayesian, priors, likelihood, covariance |
| SpRBM | visual, object, objects, images, vision | neurons, biology, spike, synaptic, realistic | conditional, probability, bayesian, hidden, mackay |
| CaRBM | image, images, pixels, objects, recognition | membrane, resting, inhibitory, physiol, excitatory | likelihood, hyperparameters, monte, variational, neal |

Table 2: Topics learned by each model on the NIPS dataset. Each column corresponds to a chosen topic, and each cell corresponds to a single hidden unit. The hidden unit is chosen as the best match to the given topic from amongst all of the hidden units learned by the model in the row.

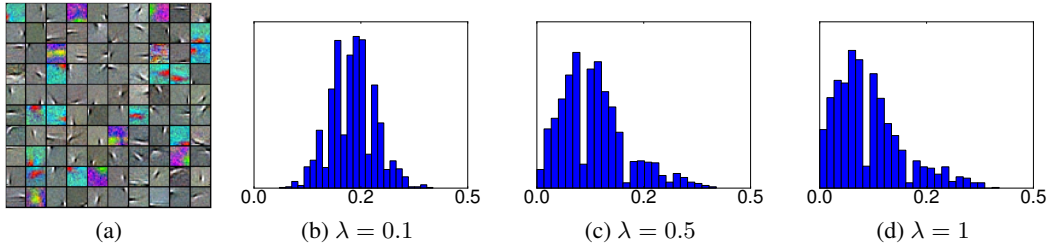

(a)  (b) $\lambda = 0.1$  (c) $\lambda = 0.5$  (d) $\lambda = 1$

Figure 2: (a) Weights of the CaRBM learned on $16 \times 16$ images patches sampled from the CIFAR-10 dataset. (b)-(c) Change in population sparsity with increasing KL penalty $\lambda$ on the NIPS dataset. The SpRBM is sensitive to $\lambda$, and can fail to model certain examples if $\lambda$ is set too high.

As it is increased, the lifetime sparsity better matches the target but at the cost of an increasing number of dead examples. This may hurt the generative performance of the SpRBM.

## 6  Conclusion

We have introduced cardinality potentials into the energy function of a Restricted Boltzmann Machine in order to enforce sparsity in the hidden representation. We showed how to use an auxiliary variable representation in order to perform efficient posterior inference and sampling. Furthermore, we showed how the marginal probabilities can be treated as nonlinearities, and how a simple finite-difference trick from Domke [14] can be used to backpropagate through the network. We found that the CaRBM performs similarly to an RBM that has been trained with a sparsity-encouraging regularizer, with the exception being datasets that exhibit a wide range of variability in the number of active inputs (e.g. text), where the SpRBM seems to have difficulty matching the target sparsity. It is possible that this effect may be significant in other kinds of data, such as images with high amounts of lighting variation.

There are a number of possible extensions to the CaRBM. For example, the cardinality potentials can be relaxed to encourage sparsity, but not enforce it, and they can be learned along with the other model parameters. It would also be interesting to see if other high order potentials could be used within the RBM framework. Finally, it would be worth exploring the use of the sparse marginal nonlinearity in auto-encoder architectures and in the deeper layers of a deep belief network.

## Footnotes

[1] The derivatives with respect to the bias terms take a similar form.

[2] http://psiexp.ss.uci.edu/research/programs_data/toolbox.htm

## References

[1] P. Smolensky. Information processing in dynamical systems: foundations of harmony theory. In *Parallel Distributed Processing: Explorations in the Microstructure of Cognition, vol. 1*, pages 194–281. MIT Press, 1986.

[2] G.E. Hinton, S. Osindero, and Y.W. Teh. A fast learning algorithm for deep belief nets. *Neural Computation*, 18(7):1527–1554, 2006.

[3] H. Lee, R. Grosse, R. Ranganath, and A.Y. Ng. Convolutional deep belief networks for scalable unsupervised learning of hierarchical representations. In *International Conference on Machine Learning*, 2009.

[4] Y. Bengio, P. Lamblin, D. Popovici, and H. Larochelle. Greedy layer-wise training of deep networks. *Advances in Neural Information Processing Systems*, 2007.

[5] J. Snoek, R. P. Adams, and H. Larochelle. Nonparametric guidance of autoencoder representations using label information. *Journal of Machine Learning Research*, 13:2567–2588, 2012.

[6] J. Yang, K. Yu, Y. Gong, and T. Huang. Linear spatial pyramid matching using sparse coding for image classification. In *Computer Vision and Pattern Recognition*, 2009.

[7] I. Goodfellow, Q. Le, A. Saxe, H. Lee, and A.Y. Ng. Measuring invariances in deep networks. *Advances in Neural Information Processing Systems*, 2009.

[8] B.A. Olshausen and D.J. Field. Sparse coding with an overcomplete basis set: A strategy employed by V1? *Vision Research*, 37(23):3311–3325, 1997.

[9] I. Goodfellow, A. Courville, and Y. Bengio. Large-scale feature learning with spike-and-slab sparse coding. *International Conference on Machine Learning*, 2012.

[10] H. Lee, C. Ekanadham, and A. Ng. Sparse deep belief net model for visual area V2. *Advances in Neural Information Processing Systems*, 2007.

[11] R. Gupta, A. Diwan, and S. Sarawagi. Efficient inference with cardinality-based clique potentials. In *International Conference on Machine Learning*, 2007.

[12] D. Tarlow, I. Givoni, and R. Zemel. HOP-MAP: Efficient message passing for high order potentials. In *Artificial Intelligence and Statistics*, 2010.

[13] M. H. Gail, J. H. Lubin, and L. V. Rubinstein. Likelihood calculations for matched case-control studies and survival studies with tied death times. *Biometrika*, 68:703–707, 1981.

[14] J. Domke. Implicit differentiation by perturbation. *Advances in Neural Information Processing Systems*, 2010.

[15] G.E. Hinton. Training products of experts by minimizing contrastive divergence. *Neural Computation*, 14(8):1771–1800, 2002.

[16] V. Nair and G.E. Hinton. 3d object recognition with deep belief nets. *Advances in Neural Information Processing Systems*, 2009.

[17] R. E. Barlow and K. D. Heidtmann. Computing k-out-of-n system reliability. *IEEE Transactions on Reliability*, 33:322–323, 1984.

[18] D. Tarlow, K. Swersky, R. Zemel, R.P. Adams, and B. Frey. Fast exact inference for recursive cardinality models. In *Uncertainty in Artificial Intelligence*, 2012.

[19] L. Belfore. An O(n) log2(n) algorithm for computing the reliability of k-out-of-n:G and k-to-l-out-of-n:G systems. *IEEE Transactions on Reliability*, 44(1), 1995.

[20] T. Tieleman. Training restricted Boltzmann machines using approximations to the likelihood gradient. In *International Conference on Machine Learning*, 2008.

[21] H.J. Kappen. Deterministic learning rules for Boltzmann machines. *Neural Networks*, 8(4):537–548, 1995.

[22] H. Larochelle, Y. Bengio, and J. Turian. Tractable multivariate binary density estimation and the restricted Boltzmann forest. *Neural Computation*, 22(9):2285–2307, 2010.

[23] G.E. Hinton and R.R. Salakhutdinov. Reducing the dimensionality of data with neural networks. *Science*, 313(5786):504–507, 2006.

[24] K. Swersky, M. Ranzato, D. Buchman, B.M. Marlin, and N. de Freitas. On autoencoders and score matching for energy based models. In *International Conference on Machine Learning*, 2011.

[25] P. Vincent. A connection between score matching and denoising autoencoders. *Neural Computation*, 23(7):1661–1674, 2011.

[26] H. Larochelle, D. Erhan, A. Courville, J. Bergstra, and Y. Bengio. An empirical evaluation of deep architectures on problems with many factors of variation. In *International Conference on Machine Learning*, 2007.

[27] G.E. Hinton. A practical guide to training restricted Boltzmann machines. Technical Report UTML-TR 2010003, Department of Computer Science, University of Toronto, 2010.

[28] J. Bergstra and Y. Bengio. Random search for hyper-parameter optimization. *The Journal of Machine Learning Research*, 13:281–305, 2012.

[29] A. Krizhevsky. Learning multiple layers of features from tiny images. Master's thesis, University of Toronto, 2009.

